# A Geometric Interpretation of $\nu$−SVM Classifiers

**David J. Crisp**
Centre for Sensor Signal and
Information Processing,
Deptartment of Electrical Engineering,
University of Adelaide, South Australia
*dcrisp@eleceng.adelaide.edu.au*

**Christopher J.C. Burges**
Advanced Technologies,
Bell Laboratories,
Lucent Technologies
Holmdel, New Jersey
*burges@lucent.com*

## Abstract

We show that the recently proposed variant of the Support Vector machine (SVM) algorithm, known as $\nu$-SVM, can be interpreted as a maximal separation between subsets of the convex hulls of the data, which we call soft convex hulls. The soft convex hulls are controlled by choice of the parameter $\nu$. If the intersection of the convex hulls is empty, the hyperplane is positioned halfway between them such that the distance between convex hulls, measured along the normal, is maximized; and if it is not, the hyperplane's normal is similarly determined by the soft convex hulls, but its position (perpendicular distance from the origin) is adjusted to minimize the error sum. The proposed geometric interpretation of $\nu$-SVM also leads to necessary and sufficient conditions for the existence of a choice of $\nu$ for which the $\nu$-SVM solution is nontrivial.

## 1 Introduction

Recently, Schölkopf et al. [1] introduced a new class of SVM algorithms, called $\nu$-SVM, for both regression estimation and pattern recognition. The basic idea is to remove the user-chosen error penalty factor $C$ that appears in SVM algorithms by introducing a new variable $\rho$ which, in the pattern recognition case, adds another degree of freedom to the margin. For a given normal to the separating hyperplane, the size of the margin increases linearly with $\rho$. It turns out that by adding $\rho$ to the primal objective function with coefficient $-\nu$, $\nu \geq 0$, the variable $C$ can be absorbed, and the behaviour of the resulting SVM - the number of margin errors and number of support vectors - can to some extent be controlled by setting $\nu$. Moreover, the decision function produced by $\nu$-SVM can also be produced by the original SVM algorithm with a suitable choice of $C$.

In this paper we show that $\nu$-SVM, for the pattern recognition case, has a clear geometric interpretation, which also leads to necessary and sufficient conditions for the existence of a nontrivial solution to the $\nu$-SVM problem. All our considerations apply to feature space, after the mapping of the data induced by some kernel. We adopt the usual notation: $w$ is the normal to the separating hyperplane, the mapped

data is denoted by $x_i \in \Re^N$, $i = 1, \cdots, l$, with corresponding labels $y_i \in \{\pm 1\}$, $b$, $\rho$ are scalars, and $\xi_i$, $i = 1, \cdots, l$ are positive scalar slack variables.

## 2   ν-SVM Classifiers

The ν-SVM formulation, as given in [1], is as follows: minimize

$$F' = \frac{1}{2}\|w'\|^2 - \nu\rho' + \frac{1}{l}\sum_i \xi_i' \tag{1}$$

with respect to $w', b', \rho', \xi_i'$, subject to:

$$y_i(w' \cdot x_i + b') \geq \rho' - \xi_i', \quad \xi_i' \geq 0, \quad \rho' \geq 0. \tag{2}$$

Here $\nu$ is a user-chosen parameter between 0 and 1. The decision function (whose sign determines the label given to a test point $x$) is then:

$$f'(x) = w' \cdot x + b'. \tag{3}$$

The Wolfe dual of this problem is: maximize $F_D' = -\frac{1}{2}\sum_{ij} \alpha_i \alpha_j y_i y_j x_i \cdot x_j$ subject to

$$0 \leq \alpha_i \leq \frac{1}{l}, \quad \sum_i \alpha_i y_i = 0, \quad \sum_i \alpha_i \geq \nu \tag{4}$$

with $w'$ given by $w' = \sum_i \alpha_i y_i x_i$. Schölkopf et al. [1] show that $\nu$ is an upper bound on the fraction of margin errors[1], a lower bound on the fraction of support vectors, and that both of these quantities approach $\nu$ asymptotically.

Note that the point $w' = b' = \rho = \xi_i' = 0$ is feasible, and that at this point, $F' = 0$. Thus any solution of interest must have $F' \leq 0$. Furthermore, if $\nu\rho' = 0$, the optimal solution is at $w' = b' = \rho = \xi_i' = 0$[2]. Thus we can assume that $\nu\rho' > 0$ (and therefore $\nu > 0$) always. Given this, the constraint $\rho' \geq 0$ is in fact redundant: a negative value of $\rho'$ cannot appear in a solution (to the problem with this constraint removed) since the above (feasible) solution (with $\rho' = 0$) gives a lower value for $F'$. Thus below we replace the constraints (2) by

$$y_i(w' \cdot x_i + b') \geq \rho' - \xi_i', \quad \xi_i' \geq 0. \tag{5}$$

### 2.1   A Reparameterization of ν–SVM

We reparameterize the primal problem by dividing the objective function $F'$ by $\nu^2/2$, the constraints (5) by $\nu$, and by making the following substitutions:

$$\mu = \frac{2}{\nu l}, \quad w = \frac{w'}{\nu}, \quad b = \frac{b'}{\nu}, \quad \rho = \frac{\rho'}{\nu}, \quad \xi_i = \frac{\xi_i'}{\nu}. \tag{6}$$

This gives the equivalent formulation: minimize

$$F = \|w\|^2 - 2\rho + \mu \sum_i \xi_i \tag{7}$$

with respect to $w, b, \rho, \xi_i$, subject to:

$$y_i(w \cdot x_i + b) \geq \rho - \xi_i, \quad \xi_i \geq 0. \tag{8}$$

If we use as decision function $f(x) \equiv f'(x)/\nu$, the formulation is exactly equivalent, although both primal and dual appear different. The dual problem is now: minimize

$$F_D = \frac{1}{4} \sum_{i,j} \alpha_i \alpha_j y_i y_j x_i \cdot x_j \tag{9}$$

with respect to the $\alpha_i$, subject to:

$$\sum_i \alpha_i y_i = 0, \quad \sum_i \alpha_i = 2, \quad 0 \leq \alpha_i \leq \mu \tag{10}$$

with $w$ given by $w = \frac{1}{2} \sum_i \alpha_i y_i x_i$. In the following, we will refer to the reparameterized version of $\nu$-SVM given above as $\mu$-SVM, although we emphasize that it describes the same problem.

## 3  A Geometric Interpretation of $\nu$−SVM

In the separable case, it is clear that the optimal separating hyperplane is just that hyperplane which bisects the shortest vector joining the convex hulls of the positive and negative polarity points[3]. We now show that this geometric interpretation can be extended to the case of $\nu$−SVM for both separable and nonseparable cases.

### 3.1  The Separable Case

We start by giving the analysis for the separable case. The convex hulls of the two classes are

$$H_+ = \left\{ \sum_{i:y_i=+1} \alpha_i x_i \;\middle|\; \sum_{i:y_i=+1} \alpha_i = 1, \quad \alpha_i \geq 0 \right\} \tag{11}$$

and

$$H_- = \left\{ \sum_{i:y_i=-1} \alpha_i x_i \;\middle|\; \sum_{i:y_i=-1} \alpha_i = 1, \quad \alpha_i \geq 0 \right\}. \tag{12}$$

Finding the two closest points can be written as the following optimization problem:

$$\min_\alpha \;\; \left\| \sum_{i:y_i=+1} \alpha_i x_i - \sum_{i:y_i=-1} \alpha_i x_i \right\|^2 \tag{13}$$

subject to:

$$\sum_{i:y_i=+1} \alpha_i = 1, \qquad \sum_{i:y_i=-1} \alpha_i = 1, \qquad \alpha_i \geq 0 \qquad (14)$$

Taking the decision boundary $\bar{f}(x) = w \cdot x + \bar{b} = 0$ to be the perpendicular bisector of the line segment joining the two closest points means that at the solution,

$$w = \frac{1}{2}\Big( \sum_{i:y_i=+1} \alpha_i x_i - \sum_{i:y_i=-1} \alpha_i x_i \Big) \qquad (15)$$

and $\bar{b} = -w \cdot p$, where

$$p = \frac{1}{2}\Big( \sum_{i:y_i=+1} \alpha_i x_i + \sum_{i:y_i=-1} \alpha_i x_i \Big). \qquad (16)$$

Thus $w$ lies along the line segment (and is half its size) and $p$ is the midpoint of the line segment. By rescaling the objective function and using the class labels $y_i = \pm 1$ we can rewrite this as[4]:

$$\min_{\alpha} \quad \|w\|^2 = \frac{1}{4} \sum_{ij} \alpha_i \alpha_j y_i y_j x_i \cdot x_j \qquad (17)$$

subject to

$$\sum_i \alpha_i y_i = 0, \qquad \sum_i \alpha_i = 2, \qquad \alpha_i \geq 0. \qquad (18)$$

The associated decision function is $\bar{f}(x) = w \cdot x + \bar{b}$ where $w = \frac{1}{2}\sum_i \alpha_i y_i x_i$, $p = \frac{1}{2}\sum_i \alpha_i x_i$ and $\bar{b} = -w.p = -\frac{1}{4}\sum_{ij}\alpha_i y_i \alpha_j x_i \cdot x_j$.

### 3.2 The Connection with ν−SVM

Consider now the two sets of points defined by:

$$H_{+\mu} = \left\{ \sum_{i:y_i=+1} \alpha_i x_i \;\Bigg|\; \sum_{i:y_i=+1} \alpha_i = 1, \quad 0 \leq \alpha_i \leq \mu \right\} \qquad (19)$$

and

$$H_{-\mu} = \left\{ \sum_{i:y_i=-1} \alpha_i x_i \;\Bigg|\; \sum_{i:y_i=-1} \alpha_i = 1, \quad 0 \leq \alpha_i \leq \mu \right\}. \qquad (20)$$

We have the following simple proposition:

**Proposition 1:** $H_{+\mu} \subset H_+$ and $H_{-\mu} \subset H_-$, and $H_{+\mu}$ and $H_{-\mu}$ are both convex sets. Furthermore, the positions of the points $H_{+\mu}$ and $H_{-\mu}$ with respect to the $x_i$ do not depend on the choice of origin.

**Proof:** Clearly, since the $\alpha_i$ defined in $H_{+\mu}$ is a subset of the $\alpha_i$ defined in $H_+$, $H_{+\mu} \subset H_+$, similarly for $H_-$. Now consider two points in $H_{+\mu}$ defined by $\alpha_1$, $\alpha_2$. Then all points on the line joining these two points can be written as $\sum_{i:y_i=+1}((1-\lambda)\alpha_{1i} + \lambda\alpha_{2i})x_i$, $0 \leq \lambda \leq 1$. Since $\alpha_{1i}$ and $\alpha_{2i}$ both satisfy $0 \leq \alpha_i \leq \mu$, so does $(1-\lambda)\alpha_{1i}+\lambda\alpha_{2i}$, and since also $\sum_{i:y_i=+1}(1-\lambda)\alpha_{1i}+\lambda\alpha_{2i} = 1$, the set $H_{+\mu}$ is convex.

The argument for $H_{-\mu}$ is similar. Finally, suppose that every $x_i$ is translated by $x_0$, i.e. $x_i \rightarrow x_i + x_0 \ \forall i$. Then since $\sum_{i:y_i=+1} \alpha_i = 1$, every point in $H_{+\mu}$ is also translated by the same amount, similarly for $H_{-\mu}$. $\square$

The problem of finding the optimal separating hyperplane between the convex sets $H_{+\mu}$ and $H_{-\mu}$ then becomes:

$$\min_{\alpha} \quad \|w\|^2 = \frac{1}{4} \sum_{ij} \alpha_i \alpha_j y_i y_j x_i \cdot x_j \tag{21}$$

subject to

$$\sum_i \alpha_i y_i = 0, \qquad \sum_i \alpha_i = 2, \qquad 0 \leq \alpha_i \leq \mu. \tag{22}$$

Since Eqs. (21) and (22) are identical to (9) and (10), we see that the $\nu$–SVM algorithm is in fact finding the optimal separating hyperplane between the convex sets $H_{+\mu}$ and $H_{-\mu}$. We note that the convex sets $H_{+\mu}$ and $H_{-\mu}$ are not simply uniformly scaled versions of $H_+$ and $H_-$. An example is shown in Figure 1.

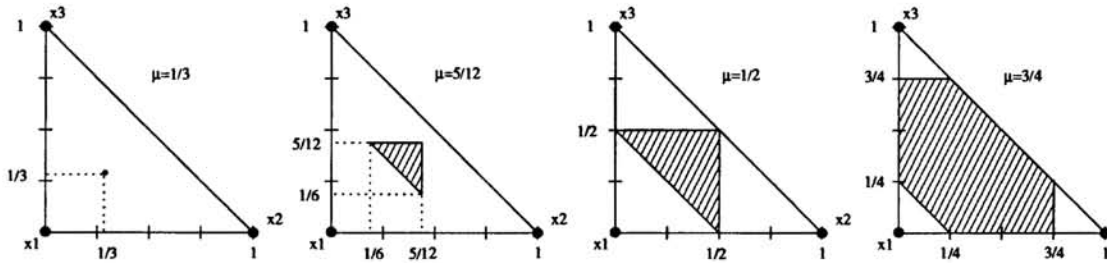

Figure 1: The soft convex hull for the vertices of a right isosceles triangle, for various $\mu$. Note how the shape changes as the set grows and is constrained by the boundaries of the encapsulating convex hull. For $\mu < \frac{1}{3}$, the set is empty.

Below, we will refer to the formulation given in this section as the soft convex hull formulation, and the sets of points defined in Eqs. (19) and (20) as soft convex hulls.

## 3.3   Comparing the Offsets and Margin Widths

The natural value of the offset $\bar{b}$ in the soft convex hull approach, $\bar{b} = -w \cdot p$, arose by asking that the separating hyperplane lie halfway between the closest extremities of the two soft convex hulls. Different choices of $b$ just amount to hyperplanes with the same normal but at different perpendicular distances from the origin. This value of $b$ will not in general be the same as that for which the cost term in Eq. (7) is minimized. We can compare the two values as follows. The KKT conditions for the $\mu$-SVM formulation are

$$(\mu - \alpha_i)\xi_i = 0 \tag{23}$$
$$\alpha_i(y_i(w \cdot x_i + b) - \rho + \xi_i) = 0 \tag{24}$$

Multiplying (24) by $y_i$, summing over $i$ and using (23) gives

$$b = \bar{b} - \frac{\mu}{2} \sum_i y_i \xi_i. \tag{25}$$

Thus the separating hyperplane found in the $\mu$-SVM algorithm sits a perpendicular distance $|\frac{\mu}{2\|w\|} \sum_i y_i \xi_i|$ away from that found in the soft convex hull formulation. For the given $w$, this choice of $b$ results in the lowest value of the cost, $\mu \sum_i \xi_i$.

The soft convex hull approach suggests taking $\bar{\rho} = w \cdot w$, since this is the value $|\tilde{f}|$ takes at the points $\sum_{y_i=+1} \alpha_i x_i$ and $\sum_{y_i=-1} \alpha_i x_i$. Again, we can use the KKT conditions to compare this with $\rho$. Summing (24) over $i$ and using (23) gives

$$\rho = \bar{\rho} + \frac{\mu}{2} \sum_i \xi_i. \tag{26}$$

Since $\bar{\rho} = w \cdot w$, this again shows that if $\rho = 0$ then $w = \xi_i = 0$, and, by (25), $b = 0$.

### 3.4 The Primal for the Soft Convex Hull Formulation

By substituting (25) and (26) into the $\mu$-SVM primal formulation (7) and (8) we obtain the primal formulation for the soft convex hull problem: minimize

$$\bar{F} = \|w\|^2 - 2\bar{\rho} \tag{27}$$

with respect to $w, \bar{b}, \bar{\rho}, \xi_i$, subject to:

$$y_i(w \cdot x_i + \bar{b}) \geq \bar{\rho} - \xi_i + \mu \sum_j \frac{1 + y_i y_j}{2} \xi_j, \qquad \xi_i \geq 0. \tag{28}$$

It is straightforward to check that the dual is exactly (9) and (10). Moreover, by summing the relevant KKT conditions, as above, we see that $\bar{b} = -w \cdot p$ and $\bar{\rho} = w \cdot w$. Note that in this formulation the variables $\xi_i$ retain their meaning according to (8).

## 4 Choosing ν

In this section we establish some results on the choices for $\nu$, using the $\mu$-SVM formulation. First, note that $\sum_i \alpha_i y_i = 0$ and $\sum_i \alpha_i = 2$ implies $\sum_{i:y_i=+1} \alpha_i = \sum_{i:y_i=-1} \alpha_i = 1$. Then $\alpha_i \geq 0$ gives $\alpha_i \leq 1$, $\forall i$. Thus choosing $\mu > 1$, which corresponds to choosing $\nu < 2/l$, results in the same solution *of the dual* (and hence the same normal $w$) as choosing $\mu = 1$. (Note that different values of $\mu > 1$ can still result in different values of the other primal variables, e.g. $b$).

The equalities $\sum_{i:y_i=+1} \alpha_i = \sum_{i:y_i=-1} \alpha_i = 1$ also show that if $\mu < 2/l$ then the feasible region for the dual is empty and hence the problem is insoluble. This corresponds to the requirement $\nu < 1$. However, we can improve upon this. Let $l_+$ ($l_-$) be the number of positive (negative) polarity points, so that $l_+ + l_- = l$. Let $l_{min} \equiv \min\{l_+, l_-\}$. Then the minimal value of $\mu$ which still results in a nonempty feasible region is $\mu_{min} = 1/l_{min}$. This gives the condition $\nu \leq 2l_{min}/l$.

We define a "nontrivial" solution of the problem to be any solution with $w \neq 0$. The following proposition gives conditions for the existence of nontrivial solutions.

**Proposition 2**: A value of $\nu$ exists which will result in a nontrivial solution to the $\nu$−SVM classification problem if and only if $\{H_{+\mu} : \mu = \mu_{min}\} \cap \{H_{-\mu} : \mu = \mu_{min}\} = \emptyset$.

**Proof**: Suppose that $\{H_{+\mu} : \mu = \mu_{min}\} \cap \{H_{-\mu} : \mu = \mu_{min}\} \neq \emptyset$. Then for all allowable values of $\mu$ (and hence $\nu$), the two convex hulls will intersect, since $\{H_{+\mu} : \mu = \mu_{min}\} \subset \{H_{+\mu} : \mu \geq \mu_{min}\}$ and $\{H_{-\mu} : \mu = \mu_{min}\} \subset \{H_{-\mu} : \mu \geq \mu_{min}\}$. If the two convex hulls intersect, then the solution is trivial, since by definition there then exist feasible points $z$ such that $z = \sum_{i:y_i=+1} \alpha_i x_i$ and $z = \sum_{i:y_i=-1} \alpha_i x_i$, and hence $2w = \sum_i \alpha_i y_i x_i = \sum_{i:y_i=+1} \alpha_i x_i - \sum_{i:y_i=-1} \alpha_i x_i = 0$ (cf. (21), (22). Now suppose that $\{H_{+\mu} : \mu = \mu_{min}\} \cap \{H_{-\mu} : \mu = \mu_{min}\} = \emptyset$. Then clearly a nontrivial solution exists, since the shortest distance between the two convex sets $\{H_{+\mu} : \mu = \mu_{min}\}$ and $\{H_{-\mu} : \mu = \mu_{min}\}$ is not zero, hence the corresponding $w \neq 0$. □

Note that when $l_+ = l_-$, the condition amounts to the requirement that the centroid of the positive examples does not coincide with that of the negative examples. Note also that this shows that, given a data set, one can find a lower bound on $\nu$, by finding the largest $\mu$ that satisfies $H_{-\mu} \cap H_{+\mu} = \emptyset$.

## 5 Discussion

The soft convex hull interpretation suggests that an appropriate way to penalize positive polarity errors differently from negative is to replace the sum $\mu \sum_i \xi_i$ in (7) with $\mu_+ \sum_{i:y_i=+1} \xi_i + \mu_- \sum_{i:y_i=-1} \xi_i$. In fact one can go further and introduce a $\mu$ for every train point. The $\mu$-SVM formulation makes this possibility explicit, which it is not in original $\nu$-SVM formulation.

Note also that the fact that $\nu$-SVM leads to values of $b$ which differ from that which would place the optimal hyperplane halfway between the soft convex hulls suggests that there may be principled methods for choosing the best $b$ for a given problem, other than that dictated by minimizing the sum of the $\xi_i$'s. Indeed, originally, the sum of $\xi_i$'s term arose in an attempt to approximate the *number* of errors on the train set [2]. The above reasoning in a sense separates the justification for $w$ from that for $b$. For example, given $w$, a simple line search could be used to find that value of $b$ which actually does minimize the number of errors on the train set. Other methods (for example, minimizing the estimated Bayes error [3]) may also prove useful.

### Acknowledgments

C. Burges wishes to thank W. Keasler, V. Lawrence and C. Nohl of Lucent Technologies for their support.

## Footnotes

[1]A margin error $x_i$ is defined to be any point for which $\xi_i > 0$ (see [1]).

[2]In fact we can prove that, even if the optimal solution is not unique, the global solutions still all have $w = 0$: see Burges and Crisp, "Uniqueness of the SVM Solution" in this volume.

[3]See, for example, K. Bennett, 1997, in http://www.rpi.edu/b̄ennek/svmtalk.ps (also, to appear).

[4]That one can rescale the objective function without changing the constraints follows from uniqueness of the solution. See also Burges and Crisp, "Uniqueness of the SVM Solution" in this volume.

### References

[1] B. Schölkopf and A. Smola and R. Williamson and P. Bartlett. New support vector algorithms, neurocolt2 nc2-tr-1998-031. Technical report, GMD First and Australian National University, 1998.

[2] C. Cortes and V. Vapnik. Support vector networks. *Machine Learning*, 20:273–297, 1995.

[3] C. J. C. Burges and B. Schölkopf. Improving the accuracy and speed of support vector learning machines. In M. Mozer, M. Jordan, and T. Petsche, editors, *Advances in Neural Information Processing Systems 9*, pages 375–381, Cambridge, MA, 1997. MIT Press.
